# Instance-Based State Identification for Reinforcement Learning

R. Andrew McCallum
Department of Computer Science
University of Rochester
Rochester, NY 14627-0226
mccallum@cs.rochester.edu

## Abstract

This paper presents *instance-based state identification*, an approach to reinforcement learning and hidden state that builds disambiguating amounts of short-term memory on-line, and also learns with an order of magnitude fewer training steps than several previous approaches. Inspired by a key similarity between learning with hidden state and learning in continuous geometrical spaces, this approach uses instance-based (or "memory-based") learning, a method that has worked well in continuous spaces.

## 1 BACKGROUND AND RELATED WORK

When a robot's next course of action depends on information that is hidden from the sensors because of problems such as occlusion, restricted range, bounded field of view and limited attention, the robot suffers from hidden state. More formally, we say a reinforcement learning agent suffers from the *hidden state problem* if the agent's state representation is non-Markovian with respect to actions and utility.

The hidden state problem arises as a case of *perceptual aliasing*: the mapping between states of the world and sensations of the agent is not one-to-one [Whitehead, 1992]. If the agent's perceptual system produces the same outputs for two world states in which different actions are required, and if the agent's state representation consists only of its percepts, then the agent will fail to choose correct actions. Note that even if an agent's state representation includes some internal state beyond its

immediate percepts, the agent can still suffer from hidden state if it does not keep *enough* internal state to uncover the non-Markovian-ness of its environment.

One solution to the hidden state problem is simply to avoid passing through the aliased states. This is the approach taken in Whitehead's *Lion* algorithm [Whitehead, 1992]. Whenever the agent finds a state that delivers inconsistent reward, it sets that state's utility so low that the policy will never visit it again. The success of this algorithm depends on a deterministic world and on the existence of a path to the goal that consists of only unaliased states.

Other solutions do not avoid aliased states, but do as best they can given a non-Markovian state representation [Littman, 1994; Singh *et al.*, 1994; Jaakkola *et al.*, 1995]. They involve either learning deterministic policies that execute incorrect actions in some aliased states, or learning stochastic policies with action choice probabilities matching the proportions of the different underlying aliased world states. These approaches do not depend on a path of unaliased states, but they have other limitations: when faced with many aliased states, a stochastic policy degenerates into random walk; when faced with potentially harmful results from incorrect actions, deterministically incorrect or probabilistically incorrect action choice may prove too dangerous; and when faced with performance-critical tasks, inefficiency that is proportional to the amount of aliasing may be unacceptable.

The most robust solution to the hidden state problem is to augment the agent's state representation on-line so as to disambiguate the aliased states. *State identification* techniques uncover the hidden state information—that is, they make the agent's internal state space Markovian. This transformation from an *imperfect state information model* to a *perfect state information model* has been formalized in the decision and control literature, and involves adding previous percepts and actions to the definition of agent internal state [Bertsekas and Shreve, 1978]. By augmenting the agent's perception with history information—short-term memory of past percepts, actions and rewards—the agent can distinguish perceptually aliased states, and can then reliably choose correct actions from them.

Predefined, fixed memory representations such as order $n$ Markov models (also known as constant-sized perception windows, linear traces or tapped-delay lines) are often undesirable. When the length of the window is more than needed, they exponentially increase the number of internal states for which a policy must be stored and learned; when the length of the memory is less than needed, the agent reverts to the disadvantages of undistinguished hidden state. Even if the agent designer understands the task well enough to know its maximal memory requirements, the agent is at a disadvantage with constant-sized windows because, for most tasks, different amounts of memory are needed at different steps of the task.

The on-line memory creation approach has been adopted in several reinforcement learning algorithms. The Perceptual Distinctions Approach [Chrisman, 1992] and Utile Distinction Memory [McCallum, 1993] are both based on splitting states of a finite state machine by doing off-line analysis of statistics gathered over many steps. Recurrent-$Q$ [Lin, 1993] is based on training recurrent neural networks. Indexed Memory [Teller, 1994] uses genetic programming to evolve agents that use load and store instructions on a register bank. A chief disadvantage of all these techniques is that they require a very large number of steps for training.

## 2 INSTANCE-BASED STATE IDENTIFICATION

This paper advocates an alternate solution to the hidden state problem we term *instance-based state identification*. The approach was inspired by the successes of instance-based (also called "memory-based") methods for learning in continuous perception spaces, (*i.e.* [Atkeson, 1992; Moore, 1992]).

The application of instance-based learning to short-term memory for hidden state is driven by the important insight that learning in continuous spaces and learning with hidden state have a crucial feature in common: they both begin learning without knowing the final granularity of the agent's state space. The former learns which regions of continuous input space can be represented uniformly and which areas must be finely divided among many states. The later learns which percepts can be represented uniformly because they uniquely identify a course of action without the need for memory, and which percepts must be divided among many states each with their own detailed history to distinguish them from other perceptually aliased world states. The first approach works with a continuous geometrical input space, the second works with a percept-action-reward "sequence" space, (or "history" space). Large continuous regions correspond to less-specified, small memories; small continuous regions correspond to more-specified, large memories.

Furthermore, learning in continuous spaces and sequence spaces both have a lot to gain from instance-based methods. In situations where the state space granularity is unknown, it is especially useful to memorize the raw previous experiences. If the agent tries to fit experience to its current, flawed state space granularity, it is bound to lose information by attributing experience to the wrong states. Experience attributed to the wrong state turns to garbage and is wasted. When faced with an evolving state space, keeping raw previous experience is the path of least commitment, and thus the most cautious about losing information.

## 3 NEAREST SEQUENCE MEMORY

There are many possible instance-based techniques to choose from, but we wanted to keep the first application simple. With that in mind, this initial algorithm is based on *k*-nearest neighbor. We call it *Nearest Sequence Memory*, (NSM). It bears emphasizing that this algorithm is the most straightforward, simple, almost naive combination of instance-based methods and history sequences that one could think of; there are still more sophisticated instance-based methods to try. The surprising result is that such a simple technique works as well as it does.

Any application of *k*-nearest neighbor consists of three parts: 1) recording each experience, 2) using some distance metric to find neighbors of the current query point, and 3) extracting output values from those neighbors. We apply these three parts to action-percept-reward sequences and reinforcement learning by *Q*-learning [Watkins, 1989] as follows:

1. For each step the agent makes in the world, it records the action, percept and reward by adding a new state to a single, long chain of states. Thus, each state in the chain contains a snapshot of immediate experience; and all the experiences are laid out in a time-connected history chain.

**Learning in a Geometric Space**
*k*-nearest neighbor, *k* = 3

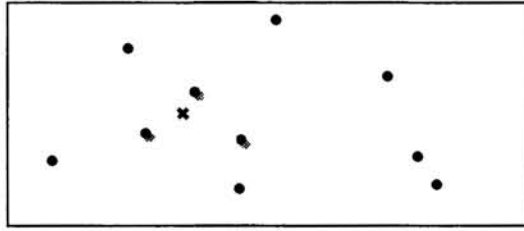

**Learning in a Sequence Space**
*k*-nearest neighbor, *k* = 3

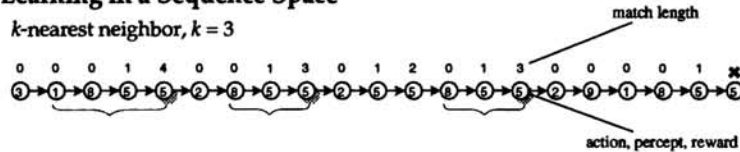

Figure 1: A continuous space compared with a sequence space. In each case, the "query point" is indicated with a gray cross, and the three nearest neighbors are indicated with gray shadows. In a geometric space, the neighborhood metric is defined by Euclidean distance. In a sequence space, the neighborhood metric is determined by sequence match length—the number of preceding states that match the states preceding the query point.

2. When the agent is about to choose an action, it finds states considered to be similar by looking in its state chain for states with histories similar to the current situation. The longer a state's string of previous experiences matches the agent's most recent experiences, the more likely the state represents where the agent is now.

3. Using the states, the agent obtains $Q$-values by averaging together the expected future reward values associated with the $k$ nearest states for each action. The agent then chooses the action with the highest $Q$-value. The regular $Q$-learning update rule is used to update the $k$ states that voted for the chosen action.

Choosing to represent short-term memory as a linear trace is a simple, well-established technique. Nearest Sequence Memory uses a linear trace to represent memory, but it differs from the *fixed-sized* window approaches because it provides a variable memory-length—like $k$-nearest neighbor, NSM can represent varying resolution in different regions of state space.

## 4   DETAILS OF THE ALGORITHM

A more complete description of Nearest Sequence Memory, its performance and its possible improvements can be found in [McCallum, 1995].

The interaction between the agent and its environment is described by actions, percepts and rewards. There is a finite set of possible actions, $\mathcal{A} = \{a_1, a_2, ..., a_m\}$,

a finite set of possible percepts, $\mathcal{O} = \{o_1, o_2, ..., o_n\}$, and scalar range of possible rewards, $\mathcal{R} = [x, y], x, y \in \Re$. At each time step, $t$, the agent executes an action, $a_t \in \mathcal{A}$, then as a result receives a new percept, $o_t \in \mathcal{O}$, and a reward, $r_t \in \mathcal{R}$. The agent records its raw experience at time $t$ in a "state" data point, $s_t$. Also associated with $s_t$ is a slot to hold a single expected future discounted reward value, denoted $q(s_t)$. This value is associated with $a_t$ and no other action.

1. Find the $k$ nearest neighbor (most similar) states for each possible future action. The state currently at the end of the chain is the "query point" from which we measure all the distances. The neighborhood metric is defined by the number of preceding experience records that match the experience records preceding the "query point" state. (Here higher values of $n(s_i, s_j)$ indicate that $s_i$ and $s_j$ are closer neighbors.)

$$n(s_i, s_j) = \begin{cases} 1 + n(s_{i-1}, s_{j-1}), & \text{if } (a_{i-1} = a_{j-1}) \wedge (o_{i-1} = o_{j-1}) \wedge (r_{i-1} = r_{j-1}) \\ 0, & \text{otherwise} \end{cases}$$

(1)

Considering each of the possible future actions in turn, we find the $k$ nearest neighbors and give them a vote, $v(s_i)$.

$$v(s_i) = \begin{cases} 1, & \text{if } n(s_t, s_i) \text{ is among the } k \max_{\forall s_j | a_j = a_i} n(s_t, s_j)\text{'s} \\ 0, & \text{otherwise} \end{cases}$$

(2)

2. Determine the $Q$-value for each action by averaging individual the $q$-values from the $k$ voting states for that action.

$$Q_t(a_i) = \sum_{\forall s_j | a_j = a_i} (v(s_i)/k)\, q(s_j)$$

(3)

3. Select an action by maximum $Q$-value, or by random exploration. According to an exploration probability, $e$, either let $a_{t+1}$ be randomly chosen from $\mathcal{A}$, or

$$a_{t+1} = \text{argmax}_a Q_t(a)$$

(4)

4. Execute the action chosen in step 3, and record the resulting experience. Do this by creating a new "state" representing the current state of the environment, and storing the action-percept-reward triple associated with it:

Increment the time counter: $t \leftarrow t + 1$. Create $s_t$; record in it $a_t, o_t, r_t$.

The agent can limit its storage and computational load by limiting the number of instances it maintains to $N$ (where $N$ is some reasonably large number). Once the agent accumulates $N$ instances, it can discard the oldest instance each time it adds a new one. This also provides a way to handle a changing environment.

5. Update the $q$-values by vote. Perform the dynamic programming step using the standard $Q$-learning rule to update those states that voted for the chosen action. Note that this actually involves performing steps 1 and 2 to get the next $Q$-values needed for calculating the utility of the agent's current state, $U_t$. (Here $\beta$ is the learning rate.)

$$U_t = \max_a Q_t(a)$$

(5)

$$(\forall s_i | a_i = a_{t-1})\ q(s_i) \leftarrow (1 - \beta v(s_i))q(s_i) + \beta v(s_i)(r_i + \gamma U_t)$$

(6)

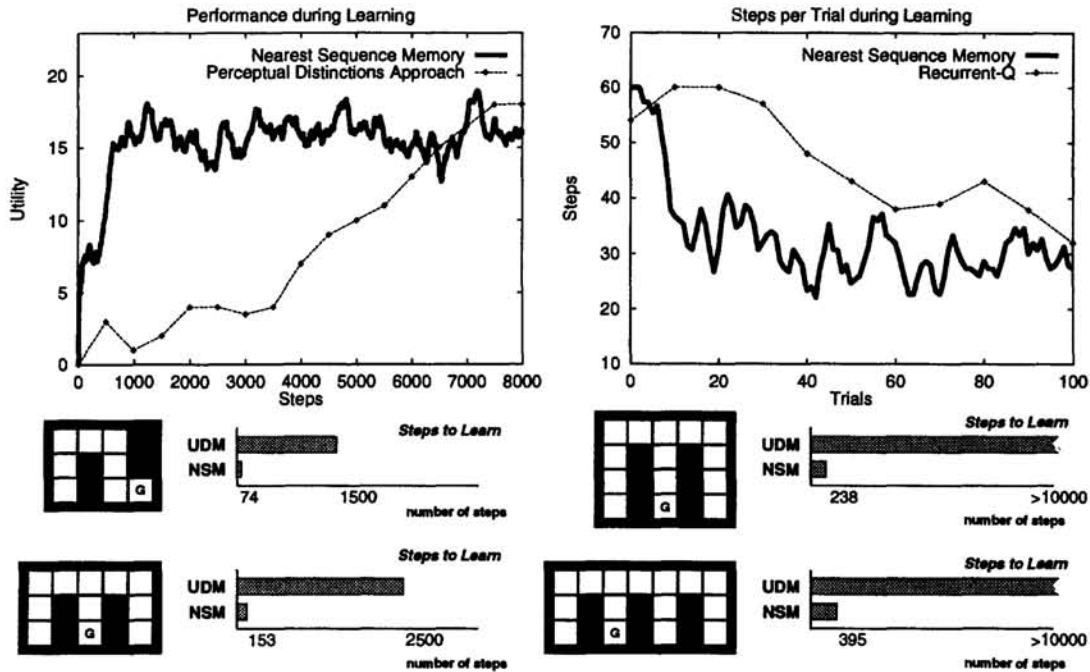

Figure 2: Comparing Nearest Sequence Memory with three other algorithms: Perceptual Distinction Approach, Recurrent-$Q$ and Utile Distinction Memory. In each case, NSM learns with roughly an order of magnitude fewer steps.

## 5   EXPERIMENTAL RESULTS

The performance of NSM is compared to three other algorithms using the tasks chosen by the other algorithms' designers. In each case, NSM learns the task with roughly an order of magnitude fewer steps. Although NSM learns good policies quickly, it does not always learn optimal policies. In section 6 we will discuss why the policies are not always optimal and how NSM could be improved.

The Perceptual Distinctions Approach [Chrisman, 1992] was demonstrated in a space ship docking application with hidden state. The task was made difficult by noisy sensors and actions. Some of the sensors returned incorrect values 30% of the time. Various actions failed 70, 30 or 20% of the time, and when they failed, resulted in random states. NSM used $\beta = 0.2$, $\gamma = 0.9$, $k = 8$, and $N = 1000$. PDA takes almost 8000 steps to learn the task. NSM learns a good policy in less than 1000 steps, although the policy is not quite optimal.

Utile Distinction Memory [McCallum, 1993] was demonstrated on several local perception mazes. Unlike most reinforcement learning maze domains, the agent perceives only four bits indicating whether there is a barrier to the immediately adjacent north, east, south and west. NSM used $\beta = 0.9$, $\gamma = 0.9$, $k = 4$, and $N = 1000$. In two of the mazes, NSM learns the task in only about 1/20th the time required by UDM; in the other two, NSM learns mazes that UDM did not solve at all.

Recurrent-$Q$ [Lin, 1993] was demonstrated on a robot 2-cup retrieval task. The environment is deterministic, but the task is made difficult by two nested levels of hidden state and by providing no reward until the task is completely finished. NSM used $\beta = 0.9$, $\gamma = 0.9$, $k = 4$, and $N = 1000$. NSM learns good performance in about 15 trials, Recurrent-$Q$ takes about 100 trials to reach equivalent performance.

## 6   DISCUSSION

Nearest Sequence Memory offers much improved on-line performance and fewer training steps than its predecessors. Why is the improvement so dramatic? I believe the chief reason lies with the inherent advantage of instance-based methods, as described in section 2: the key idea behind Instance-Based State Identification is the recognition that recording raw experience is particularly advantageous when the agent is learning a policy over a changing state space granularity, as is the case when the agent is building short-term memory for disambiguating hidden state.

If, instead of using an instance-based technique, the agent simply averages new experiences into its current, flawed state space model, the experiences will be applied to the wrong states, and cannot be reused when the agent reconfigures its state space. Furthermore, and perhaps even more detrimentally, incoming data is always interpreted in the context of the flawed state space, always biased in an inappropriate way—not simply recorded, kept uncommitted and open to easy reinterpretation in light of future data.

The experimental results in this paper bode well for instance-based state identification. Nearest Sequence Memory is simple—if such a simplistic implementation works as well as it does, more sophisticated approaches may work even better. Here are some ideas for improvement:

The agent should use a more sophisticated neighborhood distance metric than exact string match length. A new metric could account for distances between different percepts instead of considering only exact matches. A new metric could also handle continuous-valued inputs.

Nearest Sequence Memory demonstrably solves tasks that involve noisy sensation and action, but it could perhaps handle noise even better if it used some technique for explicitly separating noise from structure. $K$-nearest neighbor does not explicitly discriminate between structure and noise. If the current query point has neighbors with wildly varying output values, there is no way to know if the variations are due to noise, (in which case they should all be averaged), or due to fine-grained structure of the underlying function (in which case only the few closest should be averaged). Because NSM is built on $k$-nearest neighbor, it suffers from the same inability to methodically separate history differences that are significant for predicting reward and history differences that are not. I believe this is the single most important reason that NSM sometimes did not find optimal policies.

Work in progress addresses the structure/noise issue by combining instance-based state identification with the structure/noise separation method from Utile Distinction Memory [McCallum, 1993]. The algorithm, called *Utile Suffix Memory*, uses a tree-structured representation, and is related to work with Ron, Singer and Tishby's Prediction Suffix Trees, Moore's Parti-game, Chapman and Kaelbling's

G-algorithm, and Moore's Variable Resolution Dynamic Programming. See [Mc-Callum, 1994] for more details as well as references to this related work.

## Acknowledgments

This work has benefited from discussions with many colleagues, including: Dana Ballard, Andrew Moore, Jeff Schneider, and Jonas Karlsson. This material is based upon work supported by NSF under Grant no. IRI-8903582 and by NIH/PHS under Grant no. 1 R24 RR06853-02.

## References

[Atkeson, 1992] Christopher G. Atkeson. Memory-based approaches to approximating continuous functions. In M. Casdagli and S. Eubank, editors, *Nonlinear Modeling and Forecasting*, pages 503–521. Addison Wesley, 1992.

[Bertsekas and Shreve, 1978] Dimitri. P. Bertsekas and Steven E. Shreve. *Stochastic Optimal Control*. Academic Press, 1978.

[Chrisman, 1992] Lonnie Chrisman. Reinforcement learning with perceptual aliasing: The perceptual distinctions approach. In *Tenth Nat'l Conf. on AI*, 1992.

[Jaakkola *et al.*, 1995] Tommi Jaakkola, Satinder Pal Singh, and Michael I. Jordan. Reinforcement learning algorithm for partially observable markov decision problems. In *Advances of Neural Information Processing Systems 7*. Morgan Kaufmann, 1995.

[Lin, 1993] Long-Ji Lin. *Reinforcement Learning for Robots Using Neural Networks*. PhD thesis, Carnegie Mellon, School of Computer Science, January 1993.

[Littman, 1994] Michael Littman. Memoryless policies: Theoretical limitations and practical results. In *Proceedings of the Third International Conference on Simulation of Adaptive Behavior: From Animals to Animats*, 1994.

[McCallum, 1993] R. Andrew McCallum. Overcoming incomplete perception with utile distinction memory. In *The Proceedings of the Tenth International Machine Learning Conference*. Morgan Kaufmann Publishers, Inc., 1993.

[McCallum, 1994] R. Andrew McCallum. Utile suffix memory for reinforcement learning with hidden state. TR 549, U. of Rochester, Computer Science, 1994.

[McCallum, 1995] R. Andrew McCallum. Hidden state and reinforcement learning with instance-based state identification. *IEEE Trans. on Systems, Man, and Cybernetics*, 1995. (In press) [Earlier version available as U. of Rochester TR 502].

[Moore, 1992] Andrew Moore. *Efficient Memory-based Learning for Robot Control*. PhD thesis, University of Cambridge, November 1992.

[Singh *et al.*, 1994] Satinder Pal Singh, Tommi Jaakkola, and Michael I. Jordan. Model-free reinforcement learning for non-markovian decision problems. In *The Proceedings of the Eleventh International Machine Learning Conference*, 1994.

[Teller, 1994] Astro Teller. The evolution of mental models. In Kim Kinnear, editor, *Advances in Genetic Programming*, chapter 9. MIT Press, 1994.

[Watkins, 1989] Chris Watkins. *Learning from delayed rewards*. PhD thesis, Cambridge University, 1989.

[Whitehead, 1992] Steven Whitehead. *Reinforcement Learning for the Adaptive Control of Perception and Action*. PhD thesis, Department of Computer Science, University of Rochester, 1992.